# Learning the Semantic Correlation: An Alternative Way to Gain from Unlabeled Text

**Yi Zhang**
Machine Learning Department
Carnegie Mellon University
yizhang1@cs.cmu.edu

**Jeff Schneider**
The Robotics Institute
Carnegie Mellon University
schneide@cs.cmu.edu

**Artur Dubrawski**
The Robotics Institute
Carnegie Mellon University
awd@cs.cmu.edu

## Abstract

In this paper, we address the question of what kind of knowledge is *generally transferable* from unlabeled text. We suggest and analyze the semantic correlation of words as a generally transferable structure of the language and propose a new method to learn this structure using an appropriately chosen latent variable model. This semantic correlation contains structural information of the language space and can be used to control the joint shrinkage of model parameters for any specific task in the same space through regularization. In an empirical study, we construct 190 different text classification tasks from a real-world benchmark, and the unlabeled documents are a mixture from all these tasks. We test the ability of various algorithms to use the mixed unlabeled text to enhance all classification tasks. Empirical results show that the proposed approach is a reliable and scalable method for semi-supervised learning, regardless of the source of unlabeled data, the specific task to be enhanced, and the prediction model used.

## 1 Introduction

The availability of large amounts of unlabeled data such as text on the Internet is a strong motivation for research in semi-supervised learning [4]. Currently, most of these methods assume that the unlabeled data belong to the same classes or share the generative distributions with the labeled examples, e.g., generative models [10], low-density separation [8, 13], and graph-based methods [3]. As indicated in [11], unlabeled data in real-world applications do not necessarily follow the classes or distribution of labeled examples, and semi-supervised learning algorithms that give up this assumption have wider applicability in practice. As a result, some algorithms avoid using unlabeled examples directly in model training and instead focus on "changes of representation" that find a more informative representation from unlabeled data and use it to encode the labeled examples [4, 1, 11].

However, even algorithms for learning good features from unlabeled data still make a strong assumption: those learned high-level features will be relevant to the specific prediction task at hand. This assumption might be problematic. Many functions can be defined over an input space and a specific task corresponds to only one of them. The feature extraction on unlabeled data is an unsupervised process and thus a "blindly" learned representation might be irrelevant to a specific task, especially when the unlabeled data are not from the same task. To tackle this problem, some recent work avoids blind feature extraction by incorporating external knowledge about the task being enhanced [1]: the high-level features are learned by principal component analysis on the weights of several models, and these models are trained from some "auxiliary" tasks constructed by domain knowledge.

In this paper, we explore the possibility of extracting *generally transferable* knowledge from unlabeled text without information about the task to be enhanced. This knowledge is represented as the semantic correlation structure of the words in the text domain and is shown to be transferable among documents of different themes. This structure is extracted using a latent topic model combined with a bootstrapping procedure. The rationale is that the latent topics (or more generally, high-level features) extracted from unlabeled data might be irrelevant to a particular task, but the word distribution in these topics reveals the structural information of the language, represented by the semantic correlation among words. For any specific task defined on the same input space, this information can be used to control the joint shrinkage of model parameters through informative regularization.

The use of covariance or correlation structure has already been mentioned in transfer learning [12, 9]. A covariance structure can be transferred from a few related tasks to a target task [12] or inferred from meta-features [9]. In fact, one way to view the present work is: 1) we automatically construct a large number of diverse but meaningful "tasks" from unlabeled text without using external knowledge, where each "task" is actually extracted as a latent variable; 2) we propose to learn the semantic correlation structure of the word space from these dummy tasks and show that this structure is generally transferable regardless of the source of unlabeled data; 3) this structure can be efficiently incorporated into a broad category of prediction models via regularization, which leads to a very scalable and applicable semi-supervised learning framework.

## 2 Semantic Correlation: Transferable Structure from Unlabeled Text

### 2.1 Latent Topics and Semantic Structure

Latent topics extracted from unlabeled text might be irrelevant to a particular task, but the composition of these topics in terms of word distribution reveals information about the semantic structure of the language. Assume a latent topic model [7, 2] of the word space $\mathbf{X}$, or more generally, a latent variable model characterizing the input space $\mathbf{X}$:

$$\mathbf{x} = \mathbf{A}\mathbf{z} \tag{1}$$

where $\mathbf{x} = [\mathbf{x}_1, \mathbf{x}_2, \ldots, \mathbf{x}_p]^T$ is the $p$-dimensional vector of input variables, and $\mathbf{z} = [\mathbf{z}_1, \mathbf{z}_2, \ldots, \mathbf{z}_k]^T$ represents latent variables in the $k$-dimensional latent space $\mathbf{Z}$. $\mathbf{A}$ is a $p \times k$ matrix, representing a generative process from a probabilistic view or a projection from a deterministic view. For a latent topic model, $\mathbf{x}$ corresponds to the bag-of-words vector of a document divided by the document length, $\mathbf{z}$ is the distribution of $k$ latent topics in the document, and $\mathbf{A}$ is the distribution of $p$ words in $k$ latent topics. Various models fit in this formula including PCA, ICA, sparse coding, and non-negative matrix factorization.

Different documents have different topic distributions, $\mathbf{z}$, and thus different word distributions, $\mathbf{x}$, but $\mathbf{A}$ can be considered an invariant structure of the language. Each p-dimensional column vector of $\mathbf{A}$ denotes the word distribution in a latent topic, and serves as an "observation" in the $p$ dimensional word space, indicating the semantic roles of $p$

words in this topic. Given a large set of $k$ latent topics represented by $k$ p-dimensional vectors $\{\mathbf{a}_{(,1)}, \mathbf{a}_{(,2)}, \ldots, \mathbf{a}_{(,k)}\}$, we can define the semantic covariance of $p$ words as follows. Let $\mathbf{A}$ denote the matrix formed by treating each vector $\mathbf{a}_{(,t)}, t = 1, 2, \ldots, k$ as a column, and let $\mathbf{a}_{(i,)}$ and $\mathbf{a}_{(i,t)}$ denote a row vector and an element of this matrix, respectively. The *semantic covariance* of word $i$ and word $j$ is defined as:

$$cov_s(\mathbf{x}_i, \mathbf{x}_j) \quad = \quad \frac{1}{k}\sum_{t=1}^{k}(\mathbf{a}_{it} - \overline{\mathbf{a}}_{(i,)})(\mathbf{a}_{jt} - \overline{\mathbf{a}}_{(j,)}) \quad = \quad \frac{1}{k}\sum_{t=1}^{k}\mathbf{a}_{it}\mathbf{a}_{jt} - \overline{\mathbf{a}}_{(i,)}\overline{\mathbf{a}}_{(j,)} \quad (2)$$

where $\overline{\mathbf{a}}_{(i,)}$ is the mean of the $i$th row in $\mathbf{A}$. Naturally, the *semantic correlation* is:

$$corr_s(\mathbf{x}_i, \mathbf{x}_j) = \frac{cov_s(\mathbf{x}_i, \mathbf{x}_j)}{\sqrt{cov_s(\mathbf{x}_i, \mathbf{x}_i)cov_s(\mathbf{x}_j, \mathbf{x}_j)}} \quad (3)$$

## 2.2  Comparing Semantic Correlation and Data Correlation

Suppose we observe a set of $n$ documents in word space $\mathbf{X}$, denoted by an $n \times p$ data matrix $\mathbf{D}_X$ where each document corresponds to a p-dimensional bag-of-words vector of counts. We refer to the correlation between words computed directly from $\mathbf{D}_X$ as the *data correlation*. This data correlation may not be transferable between tasks since documents from different themes may have distinct topic distributions and word distributions, which lead to different word correlations in data space.

Here we show intuitively why we expect the data correlation to have limited use across distinct tasks, while we expect the semantic correlation to be transferable. Consider the latent variable model in eq. (1), which relates $\mathbf{A}$ to data space $\mathbf{X}$. We focus on semantic covariance and data covariance, and assume that the bag-of-words vector is divided by the length of the document so that it corresponds to $\mathbf{x}$ in eq. (1). From eq. (1), an input variable $\mathbf{x}_i$ can be written as $\mathbf{x}_i = \sum_{t=1}^{k} \mathbf{a}_{it}\mathbf{z}_t$, and therefore, the data covariance of word $i$ and word $j$ can be expressed as:

$$cov(\mathbf{x}_i, \mathbf{x}_j) = E[(\mathbf{x}_i - E\mathbf{x}_i)(\mathbf{x}_j - E\mathbf{x}_j)] \quad (4)$$
$$= E[\sum_{t=1}^{k}\mathbf{a}_{it}(\mathbf{z}_t - E\mathbf{z}_t)\sum_{t=1}^{k}\mathbf{a}_{jt}(\mathbf{z}_t - E\mathbf{z}_t)]$$
$$= \sum_{t=1}^{k}\sum_{t'=1}^{k}\mathbf{a}_{it}\mathbf{a}_{jt'} E[(\mathbf{z}_t - E\mathbf{z}_t)(\mathbf{z}_{t'} - E\mathbf{z}_{t'})]$$
$$= \sum_{t=1}^{k}\sum_{t'=1}^{k}\mathbf{a}_{it}\mathbf{a}_{jt'} cov(\mathbf{z}_t, \mathbf{z}_{t'})$$

Thus, data covariance is directly related to the covariance among latent topics. Documents from different sources have different topic distributions and thus different covariance terms $cov(\mathbf{z}_t, \mathbf{z}_{t'})$ in latent space. As a result, the data covariance learned from one source of documents may *not* be transferable to another class of documents. On the other hand, the semantic covariance in eq. (2) is completely determined by the structure of $\mathbf{A}$.

Intuitively, the data covariance among words must contain some information about the semantic relationship of words. This can also be observed from eq. (4). If we ignore the effect of the covariance among topics by assuming that latent topics are independently distributed and have the same variance (denoted as $\sigma^2$), eq. (4) can be written as:

$$cov(\mathbf{x}_i, \mathbf{x}_j) \quad = \quad \sigma^2\sum_{t=1}^{k}\mathbf{a}_{it}\mathbf{a}_{jt} \quad (5)$$

---

**Algorithm 1** Estimation of semantic correlation structure

---
**Input:** data $D = D_u \cup D_l$, latent variable model $M$
**Output:** semantic correlation matrix $\mathbf{\Sigma}_s$
**Parameters:** $\alpha$, $k$, $N$
Initialize $\mathbf{V} \leftarrow \emptyset$
**repeat**
   $D_{samp} \leftarrow Sampling(D, \alpha)$
   $\{(\mathbf{z}_1, \mathbf{a}_{(,1)}), (\mathbf{z}_2, \mathbf{a}_{(,2)}), \ldots, (\mathbf{z}_k, \mathbf{a}_{(,k)})\} \leftarrow M(k, D_{samp})$
   $\mathbf{V} \leftarrow \mathbf{V} \cup \{\mathbf{a}_{(,1)}, \mathbf{a}_{(,2)}, \ldots, \mathbf{a}_{(,k)}\}$
**until** $|\mathbf{V}| \geq kN$
Compute $\mathbf{\Sigma}_s$: $\mathbf{\Sigma}_s(i, j) \leftarrow corr_s(\mathbf{x}_i, \mathbf{x}_j)$

---

Comparing this to the last form in eq. (2), we see the similarity between data and semantic covariance. In fact, our empirical study shows that data correlation from unlabeled text does contain useful information, but is not as informative as semantic correlation.

## 3 Semantic Structure Learning and Informative Regularization

Consider a set of $n_l$ labeled documents $D_l = \{(\mathbf{x}_i^l, y_i^l) \in X \times Y_l, i = 1, \cdots n_l\}$, where $X \subseteq R^p$ is the $p$-dimensional word space, and $Y_l = \{-1, 1\}$ for classification and $Y_l \subseteq R$ for regression. Also assume that a large set of $n_u$ unlabeled documents $D_u = \{\mathbf{x}_i^u \in X, i = 1, \cdots n_u\}$ is available. The goal is to learn a good function $f_l : X \rightarrow Y_l$, which is a classifier or a regressor. In this section we introduce a framework to transfer knowledge from unlabeled text. Section 3.1 proposes an approach to learning the semantic structure of the word space from a set of unlabeled text. In section 3.2, we discuss how to efficiently apply the learned structure to a broad category of prediction models through regularization.

### 3.1 Learning the Semantic Correlation

The semantic correlation among words can be estimated using eq. (3) by observing a large number of different latent topics. However, obtaining a large set of diverse but meaningful topics is hard, since the number of meaningful topics extracted by a latent topic model is usually not very large. To solve this problem, resampling techniques such as bootstrapping [5] can be combined with a chosen latent variable model, which provides a principled way to estimate the semantic correlation. The procedure is given in Algorithm 1, which uses all the available data $D = D_u \cup D_l$ and a latent variable model $M$ as the input. The algorithm repeats $N$ iterations. In each iteration it draws an $\alpha$ percentage sample[1] from the data and extracts $k$ latent topics from the sample by applying the model $M$. After $N$ iterations, the $p \times p$ semantic correlation matrix $\mathbf{\Sigma}_s$ is estimated from the $kN$ observations of word distribution in latent topics. The algorithm requires an appropriate latent variable model $M$ (e.g., latent dirichlet allocation for text data), and a number $k$ of latent variables extracted each iteration from the sampled data. The number of iterations $N$ is set as large as necessary to obtain a reliable estimation.

### 3.2 Knowledge Transfer by Informative Regularization

This section discusses how to use the semantic structure $\mathbf{\Sigma}_s$ in any specific learning task defined on the input space $X$. For the prediction model, we mainly consider regularized linear models with an $l$-2 norm penalty, e.g., support vector machines, ridge regression, logistic regression with a Gaussian prior, etc. The model is represented by a $p$-dimensional weight vector $\mathbf{w}$ and an intercept $b$. The prediction is computed as $\mathbf{w}^T \mathbf{x} + b$ for regression

or by setting a threshold $\theta$ (usually $\theta = 0$) on $\mathbf{w}^T\mathbf{x} + b$ for classification. To learn $\mathbf{w}$ and $b$, we minimize a loss function $L$ on the training examples plus a regularization term on $\mathbf{w}$:

$$\underset{\mathbf{w},b}{\operatorname{argmin}} \sum_{i=1}^{n_l} L(y_i^l, \mathbf{w}^T\mathbf{x}_i^l + b) + \lambda\mathbf{w}^T\mathbf{w} \tag{6}$$

Different models correspond to different loss functions [6], e.g., SVMs use hinge loss, logistic regression uses log-likelihood loss, and ridge regression uses squared error loss. The regularization term $\lambda\mathbf{w}^T\mathbf{w} = \lambda\mathbf{w}^T\mathbf{I}^{-1}\mathbf{w}$ is well known to be equivalent to the Bayesian approach that imposes a Gaussian prior with zero mean and an identity correlation matrix. The correlation is often set to an identity matrix due to lack of knowledge about the input space. If a covariance or correlation structure is known, e.g., the semantic structure of the word space, the prior can be more informative [12]. Incorporating $\mathbf{\Sigma}_s$ into the Gaussian prior leads to a new regularization term and the resulting model is:

$$\underset{\mathbf{w},b}{\operatorname{argmin}} \sum_{i=1}^{n_l} L(y_i^l, \mathbf{w}^T\mathbf{x}_i^l + b) + \lambda\mathbf{w}^T\mathbf{\Sigma}_s^{-1}\mathbf{w} \tag{7}$$

Extending the discussion on SVMs in [9], all regularized linear models in the form of eq. (7) can be easily solved by three steps. First, transform the training examples by

$$\tilde{\mathbf{x}}_i^l = \mathbf{\Sigma}_s^{\frac{1}{2}}\mathbf{x}_i^l \tag{8}$$

Second, learn the standard linear model in the transformed space:

$$\underset{\tilde{\mathbf{w}},b}{\operatorname{argmin}} \sum_{i=1}^{n_l} L(y_i^l, \tilde{\mathbf{w}}^T\tilde{\mathbf{x}}_i^l + b) + \lambda\tilde{\mathbf{w}}^T\tilde{\mathbf{w}} \tag{9}$$

Finally, the optimal solution for (7) is obtained by:

$$\mathbf{w} = \mathbf{\Sigma}_s^{\frac{1}{2}}\tilde{\mathbf{w}} \tag{10}$$

This equivalence is derived from $\mathbf{w}^T\mathbf{x}_i^l = \tilde{\mathbf{w}}^T\tilde{\mathbf{x}}_i^l$ and $\mathbf{w}^T\mathbf{\Sigma}_s^{-1}\mathbf{w} = \tilde{\mathbf{w}}^T\tilde{\mathbf{w}}$. Semantic correlation is transferable to any specific task and thus can be computed offline. As a result, semi-supervised learning for any task simply requires the linear transformation in eq. (8) before training on the labeled examples, which is very scalable.

## 4  Experiments

We use the by-date version of the 20-NewsGroups data set[2], where $11314$ training and $7532$ testing documents are divided by date and denoted as $D_{tr}$ and $D_{ts}$ here. Documents are represented by bag-of-words vectors. The vocabulary is built to include the most frequent $200$ words in each of the 20 newsgroups, while the 20 most frequent words over all 20 newsgroups are removed. This yields an input space $X$ with $p = 1443$ features (words).

Documents come from 20 newsgroups, so we construct 190 binary classification tasks, one for each pair of newsgroups. For each task, a few documents in the two newsgroups are selected from $D_{tr}$ as the labeled examples, denoted as $D_l$ in section 3. The rest of the documents in $D_{tr}$ are used as the unlabeled data, denoted by $D_u$. Note that $D_u$ is a *mixture* from all the 20 newsgroups. In this sense, semi-supervised learning algorithms that assume the unlabeled data come from the target task or the same generative distribution are unlikely to work very well. The test data for each binary task are all the relevant documents in $D_{ts}$, i.e., documents in $D_{ts}$ that belong to one of the two chosen newsgroups. For any task we

always have $D_u \cup D_l = D_{tr}$, so Algorithm 1 is run only once on $D_{tr}$ to learn the semantic correlation structure $\mathbf{\Sigma}_s$ that is used by all 190 tasks.

The documents are well distributed over the 20 newsgroups and thus there are large numbers of training documents in $D_{tr}$ for each newsgroup. To limit the number of labeled examples for each binary prediction task, we use $5\%, 10\%, 20\%$ of the relevant documents in $D_{tr}$ as the labeled examples $D_l$, and the rest of the relevant and all irrelevant documents in $D_{tr}$ as the unlabeled data $D_u$. We denote these tests as $5\%$-Test, $10\%$-Test, and $20\%$-Test. The result of each test is averaged over 10 random runs, with $D_l$ randomly selected from $D_{tr}$. The testing data for each task are fixed to be all relevant documents in $D_{ts}$, which is invariant for a task among different tests and random runs. Methods for comparison are as follows.

**(1) Comparison based on SVM.** For each classification task, we compare: SVM directly trained on labeled examples $D_l$ (denoted $SVM$), SVM trained on $D_l$ in the latent topic space extracted by latent dirichlet allocation on $D_l \cup D_u$ [2] (denoted $SVM_{LDA}$), SVM trained on $D_l$ in principal component space extracted by PCA on $D_l \cup D_u$ (denoted $SVM_{PCA}$), SVM trained on $D_l$ via informative regularization with *semantic* correlation $\mathbf{\Sigma}_s$ in the prior (denoted $SVM_{IR}$), SVM trained on $D_l$ via informative regularization with *data* correlation in the prior (denoted $SVM_{IR(data)}$), where the data correlation $\mathbf{\Sigma}$ is estimated from bag-of-words vectors of documents in $D_l \cup D_u$.

**(2) Comparison based on L-2 Regularized Logistic Regression.** Analogous to the SVM comparison with logistic regression (denoted LGR) as the base classifier.

**(3) Comparison based on ridge regression.** Ridge regression (denoted RR) is used as the base classifier: examples are labeled as $+1$ and $-1$, and prediction is made by $\mathbf{w}^T\mathbf{x}+b > 0$.

**(4) Comparison to semi-supervised SVM.** Recently a fast semi-supervised SVM using L-2 loss was proposed [13], which makes it possible to handle large-scale unlabeled documents. We compare: L2-SVM directly trained on $D_l$ ($L2\text{-}SVM$), semi-supervised L2-SVM trained on $D_l \cup D_u$ ($L2\text{-}S^3VM$), and L2-SVM trained on $D_l$ via informative regularization with semantic correlation ($L2\text{-}SVM_{IR}$). The semi-supervised SVM should not work well since the unlabeled data is a mixture from all tasks. Therefore, we also test an "oracle" semi-supervised SVM, using labeled examples together with unlabeled examples coming only from the two relevant newsgroups ($L2\text{-}S^3VM_{oracle}$).

Here are additional implementation details. The regularization parameter $\lambda$ for each model is determined by 5-fold cross-validation in the range $10^{-6}$ to $10^6$. LibSVM $2.85$ is used for SVM. For PCA, we tried $10, 20, 30, 50, 100, 200, 400$ principal components and report PCA using 200 principal components as the best result. For latent dirichlet allocation, we use the implementation at http://chasen.org/$\sim$daiti-m/dist/lda/. We tried $k = 10, 20, 30, 50, 100, 200$ latent topics with 30 topics performing best. For the proposed method, Algorithm 1 uses latent dirichlet allocation with $k = 30$ topics per sampling, repeats $N = 100$ iterations, and $\mathbf{\Sigma}_s$ is estimated from these 3000 latent topics. $L2\text{-}S^3VM$ (code available as SVMlin [13]) has a second parameter $\lambda_u$ for unlabeled examples, which is set to 1 as in [13]. Unlabeled data for $L2\text{-}S^3VM$ is downsampled to 3000 documents for each run to make training (and cross-validation) feasible.

Empirical results are shown in Tables 1- 4. For each semi-supervised learning algorithm, we report two performance measures: the average classification error over all 190 tasks, and the gain/loss ratio compared to the corresponding supervised learning method. The former measures the effectiveness of using the unlabeled data, while the latter measures the reliability of the knowledge transfer. From Tables 1 - 3, IR based methods with semantic correlation significantly outperform standard supervised learning, LDA based methods, PCA based methods, and is also generally more effective than IR with data correlation. The LDA based algorithms slightly improve the prediction performance when using SVM or logistic regression as the base classifier, while decreasing the performance when using ridge

Table 1: Comparison over 190 tasks, based on SVMs

|  | 5%-Test | 10%-Test | 20%-Test |
|---|---|---|---|
| $SVM$ | 14.22% | 10.34% | 7.88% |
| $SVM_{LDA(30)}$ | 9.76% (179/11) | 8.01% (171/19) | 6.90% (161/29) |
| $SVM_{PCA(200)}$ | 13.32% (123/67) | 10.31% (104/86) | 8.29% (89/101) |
| $SVM_{IR}$ | **7.58% (190/0)** | **6.11% (190/0)** | **5.13% (183/7)** |
| $SVM_{IR(data)}$ | 9.40% (185/5) | 7.14% (183/7) | 5.70% (180/10) |

Table 2: Comparison over 190 tasks, based on regularized logistic regression

|  | 5%-Test | 10%-Test | 20%-Test |
|---|---|---|---|
| $LGR$ | 11.70% | 8.43% | 6.67% |
| $LGR_{LDA(30)}$ | 8.21% (171/19) | 7.38% (156/34) | 6.79% (134/56) |
| $LGR_{PCA(200)}$ | 11.43% (105/85) | 8.95% (65/125) | 7.28% (64/122) |
| $LGR_{IR}$ | **6.70% (189/1)** | **5.78% (181/9)** | **5.19% (169/21)** |
| $LGR_{IR(data)}$ | 8.46% (172/18) | 7.21% (157/33) | 6.46% (132/58) |

Table 3: Comparison over 190 tasks, based on ridge regression

|  | 5%-Test | 10%-Test | 20%-Test |
|---|---|---|---|
| $RR$ | 14.13% | 10.73% | 8.90% |
| $RR_{LDA(30)}$ | 14.08% (111/101) | 11.98% (67/102) | 11.34% (42/148) |
| $RR_{PCA(200)}$ | 15.50% (56/132) | 12.80% (33/157) | 11.53% (17/173) |
| $RR_{IR}$ | **10.55% (182/8)** | **8.88% (161/29)** | **8.01% (134/56)** |
| $RR_{IR(data)}$ | **10.68% (176/14)** | **8.94% (157/33)** | **7.99% (139/51)** |

Table 4: Comparison to semi-supervised SVMs over 190 tasks, based on L2-SVM

|  | 5%-Test | 10%-Test | 20%-Test |
|---|---|---|---|
| $L2\text{-}SVM$ | 11.18% | 8.41% | 6.65% |
| $L2\text{-}S^3VM$ | 14.14% (14/176) | 11.64% (5/185) | 10.04% (1/189) |
| $L2\text{-}S^3VM_{oracle}$ | 8.22% (189/1) | 6.95% (185/5) | 6.00% (164/24) |
| $L2\text{-}SVM_{IR}$ | **6.87% (188/2)** | **5.73% (180/10)** | **4.98% (177/13)** |

regression. This is possibly because the loss function of ridge regression is not a good approximation to the 0/1 classification error, and therefore, ridge regression is more sensitive to irrelevant latent features extracted from mixed unlabeled documents. The PCA based methods are generally worse than standard supervised learning, which indicates they are sensitive to the mixed unlabeled data. In Table 4, the $L2\text{-}S^3VM$ performs worse than standard $L2\text{-}SVM$, showing that traditional semi-supervised learning cannot handle unlabeled data outside the target task. We can also see that the $L2\text{-}SVM_{IR}$ even outperforms the oracle version of semi-supervised SVM ($L2\text{-}S^3VM_{oracle}$) by achieving similar gain/loss ratio but better average classification error. This is a very promising result since it shows that information can be gained from other tasks even in excess of what can be gained from a significant amount of unlabeled data on the task at hand. In conclusion, the empirical results show that the proposed approach is an effective and reliable (also scalable) method for semi-supervised learning, regardless of the source of unlabeled data, the specific task to be enhanced, and the base prediction model used.

It is interesting to directly compare the semantic correlation $\Sigma_s$ and the data correlation $\Sigma$ matrices learned from the data. We make three observations: 1) The average value of entries is $0.0147$ in the semantic correlation and $0.0341$ in the data correlation. We

Table 5: Top 10 distinct word pairs in terms of semantic correlation vs. data correlation

| gaza/lebanes | biker/yamaha | motorcycl/yamaha | batter/clemen | yanke/catcher |
|---|---|---|---|---|
| 0.956/0.007 | 0.937/−0.004 | 0.970/0.030 | 0.932/−0.002 | 0.934/0.002 |
| palestin/lebanes | cage/ama | toyota/mileag | mileag/mustang | brave/batter |
| 0.946/0.181 | 0.921/−0.005 | 0.934/0.009 | 0.923/−0.002 | 0.950/0.025 |

have $1617834$ entries with higher data correlation and $462972$ entries with higher semantic correlation. Thus overall word pairs tend to have higher values in the data correlation. 2) However, if we list the top $1000$ pairs of words with the largest *absolute* difference between the two correlations, they *all* have very high semantic correlation and low data correlation. 3) We list the top 10 such word pairs and their semantic/data correlations in Table 5. The words are indeed quite related. In conclusion, entries in $\Sigma_s$ seem to have a power-law distribution where a few pairs of words have very high correlation and the rest have low correlation, which is consistent with our intuition about words. However, the data correlation misses highly correlated words found by the semantic correlation even though it generally assigns higher correlation to most word pairs. This is consistent with the data correlation not being transferable among documents of different themes. When the unlabeled documents are a mixture from different sources, the estimation of data correlation is affected by the fact that the mixture of input documents is not consistent.

## Acknowledgments

This work was supported by the Centers of Disease Control and Prevention (award R01-PH 000028) and by the National Science Foundation (grant IIS-0325581).

## Footnotes

[1]In this paper, we use $\alpha = 50\%$ sampling without replacement. Other choices can be made.

[2]http://people.csail.mit.edu/jrennie/20Newsgroups/

# References

[1] R. K. Ando and T. Zhang. A framework for learning predictive structures from multiple tasks and unlabeled data. *JMLR*, 6:1817–1853, 2005.

[2] D. M. Blei, A. Y. Ng, and M. I. Jordan. Latent dirichlet allocation. *JMLR*, 3:993–1022, 2003.

[3] A. Blum and S. Chawla. Learning from labeled and unlabeled data using graph mincuts. In *ICML*, pages 19–26, 2001.

[4] O. Chapelle, B. Scholkopf, and A. Zien. *Semi-supervised Learning*. The MIT Press, 2006.

[5] B. Efron. Bootstrap methods: Another look at the jackknife. *The Annals of Statistics*, 7, 1979.

[6] T. Hastie, R. Tibshirani, and J. Friedman. *The Elements of Statistical Learning: Data Mining, Inference and Prediction*. Springer, New York, 2001.

[7] T. Hofmann. Probabilistic latent semantic analysis. In *UAI*, 1999.

[8] T. Joachims. Transductive inference for text classification using support vector machines. In *ICML*, pages 200–209, 1999.

[9] E. Krupka and N. Tishby. Incorporating Prior Knowledge on Features into Learning. In *AIS-TATS*, pages 227–234, 2007.

[10] K. Nigam, A. K. McCallum, S. Thrun, and T. Mitchell. Text classification from labeled and unlabeled documents using em. *Machine Learning*, 39:103–134, 2000.

[11] R. Raina, A. Battle, H. Lee, and B. P. A. Y. Ng. Self-taught learning: Transfer learning from unlabeled data. In *ICML*, pages 759–766, 2007.

[12] R. Raina, A. Y. Ng, and D. Koller. Constructing informative priors using transfer learning. In *ICML*, pages 713–720, 2006.

[13] V. Sindhwani and S. Keerthi. Large scale semi-supervised linear svms. In *SIGIR*, 2006.
